# Hidden Technical Debt in Machine Learning Systems

**D. Sculley, Gary Holt, Daniel Golovin, Eugene Davydov, Todd Phillips**
`{dsculley,gholt,dgg,edavydov,toddphillips}@google.com`
Google, Inc.

**Dietmar Ebner, Vinay Chaudhary, Michael Young, Jean-François Crespo, Dan Dennison**
`{ebner,vchaudhary,mwyoung,jfcrespo,dennison}@google.com`
Google, Inc.

## Abstract

Machine learning offers a fantastically powerful toolkit for building useful complex prediction systems quickly. This paper argues it is dangerous to think of these quick wins as coming for free. Using the software engineering framework of *technical debt*, we find it is common to incur massive ongoing maintenance costs in real-world ML systems. We explore several ML-specific risk factors to account for in system design. These include boundary erosion, entanglement, hidden feedback loops, undeclared consumers, data dependencies, configuration issues, changes in the external world, and a variety of system-level anti-patterns.

## 1 Introduction

As the machine learning (ML) community continues to accumulate years of experience with live systems, a wide-spread and uncomfortable trend has emerged: developing and deploying ML systems is relatively fast and cheap, but maintaining them over time is difficult and expensive.

This dichotomy can be understood through the lens of *technical debt*, a metaphor introduced by Ward Cunningham in 1992 to help reason about the long term costs incurred by moving quickly in software engineering. As with fiscal debt, there are often sound strategic reasons to take on technical debt. Not all debt is bad, but all debt needs to be serviced. Technical debt may be paid down by refactoring code, improving unit tests, deleting dead code, reducing dependencies, tightening APIs, and improving documentation [8]. The goal is *not* to add new functionality, but to enable future improvements, reduce errors, and improve maintainability. Deferring such payments results in compounding costs. Hidden debt is dangerous because it compounds silently.

In this paper, we argue that ML systems have a special capacity for incurring technical debt, because they have all of the maintenance problems of traditional code plus an additional set of ML-specific issues. This debt may be difficult to detect because it exists at the *system* level rather than the code level. Traditional abstractions and boundaries may be subtly corrupted or invalidated by the fact that data influences ML system behavior. Typical methods for paying down code level technical debt are not sufficient to address ML-specific technical debt at the system level.

This paper does not offer novel ML algorithms, but instead seeks to increase the community's awareness of the difficult tradeoffs that must be considered in practice over the long term. We focus on system-level interactions and interfaces as an area where ML technical debt may rapidly accumulate. At a system-level, an ML model may silently erode abstraction boundaries. The tempting re-use or chaining of input signals may unintentionally couple otherwise disjoint systems. ML packages may be treated as black boxes, resulting in large masses of "glue code" or calibration layers that can lock in assumptions. Changes in the external world may influence system behavior in unintended ways. Even monitoring ML system behavior may prove difficult without careful design.

## 2 Complex Models Erode Boundaries

Traditional software engineering practice has shown that strong abstraction boundaries using encapsulation and modular design help create maintainable code in which it is easy to make isolated changes and improvements. Strict abstraction boundaries help express the invariants and logical consistency of the information inputs and outputs from an given component [8].

Unfortunately, it is difficult to enforce strict abstraction boundaries for machine learning systems by prescribing specific intended behavior. Indeed, ML is required in exactly those cases when *the desired behavior cannot be effectively expressed in software logic without dependency on external data*. The real world does not fit into tidy encapsulation. Here we examine several ways that the resulting erosion of boundaries may significantly increase technical debt in ML systems.

**Entanglement.** Machine learning systems mix signals together, entangling them and making isolation of improvements impossible. For instance, consider a system that uses features $\mathbf{x}_1, ... \mathbf{x}_n$ in a model. If we change the input distribution of values in $\mathbf{x}_1$, the importance, weights, or use of the remaining $n-1$ features may all change. This is true whether the model is retrained fully in a batch style or allowed to adapt in an online fashion. Adding a new feature $\mathbf{x}_{n+1}$ can cause similar changes, as can removing any feature $\mathbf{x}_j$. No inputs are ever really independent. We refer to this here as the CACE principle: Changing Anything Changes Everything. CACE applies not only to input signals, but also to hyper-parameters, learning settings, sampling methods, convergence thresholds, data selection, and essentially every other possible tweak.

One possible mitigation strategy is to isolate models and serve ensembles. This approach is useful in situations in which sub-problems decompose naturally such as in disjoint multi-class settings like [14]. However, in many cases ensembles work well because the errors in the component models are uncorrelated. Relying on the combination creates a strong entanglement: improving an individual component model may actually make the system accuracy worse if the remaining errors are more strongly correlated with the other components.

A second possible strategy is to focus on detecting changes in prediction behavior as they occur. One such method was proposed in [12], in which a high-dimensional visualization tool was used to allow researchers to quickly see effects across many dimensions and slicings. Metrics that operate on a slice-by-slice basis may also be extremely useful.

**Correction Cascades.** There are often situations in which model $m_a$ for problem $A$ exists, but a solution for a slightly different problem $A'$ is required. In this case, it can be tempting to learn a model $m'_a$ that takes $m_a$ as input and learns a small correction as a fast way to solve the problem.

However, this correction model has created a new system dependency on $m_a$, making it significantly more expensive to analyze improvements to that model in the future. The cost increases when correction models are cascaded, with a model for problem $A''$ learned on top of $m'_a$, and so on, for several slightly different test distributions. Once in place, a correction cascade can create an improvement deadlock, as improving the accuracy of any individual component actually leads to system-level detriments. Mitigation strategies are to augment $m_a$ to learn the corrections directly within the same model by adding features to distinguish among the cases, or to accept the cost of creating a separate model for $A'$.

**Undeclared Consumers.** Oftentimes, a prediction from a machine learning model $m_a$ is made widely accessible, either at runtime or by writing to files or logs that may later be consumed by other systems. Without access controls, some of these consumers may be *undeclared*, silently using the output of a given model as an input to another system. In more classical software engineering, these issues are referred to as visibility debt [13].

Undeclared consumers are expensive at best and dangerous at worst, because they create a hidden tight coupling of model $m_a$ to other parts of the stack. Changes to $m_a$ will very likely impact these other parts, potentially in ways that are unintended, poorly understood, and detrimental. In practice, this tight coupling can radically increase the cost and difficulty of making any changes to $m_a$ at all, even if they are improvements. Furthermore, undeclared consumers may create hidden feedback loops, which are described more in detail in section 4.

Undeclared consumers may be difficult to detect unless the system is specifically designed to guard against this case, for example with access restrictions or strict service-level agreements (SLAs). In the absence of barriers, engineers will naturally use the most convenient signal at hand, especially when working against deadline pressures.

## 3 Data Dependencies Cost More than Code Dependencies

In [13], *dependency debt* is noted as a key contributor to code complexity and technical debt in classical software engineering settings. We have found that *data dependencies* in ML systems carry a similar capacity for building debt, but may be more difficult to detect. Code dependencies can be identified via static analysis by compilers and linkers. Without similar tooling for data dependencies, it can be inappropriately easy to build large data dependency chains that can be difficult to untangle.

**Unstable Data Dependencies.** To move quickly, it is often convenient to consume signals as input features that are produced by other systems. However, some input signals are *unstable*, meaning that they qualitatively or quantitatively change behavior over time. This can happen implicitly, when the input signal comes from another machine learning model itself that updates over time, or a data-dependent lookup table, such as for computing TF/IDF scores or semantic mappings. It can also happen explicitly, when the engineering ownership of the input signal is separate from the engineering ownership of the model that consumes it. In such cases, updates to the input signal may be made at any time. This is dangerous because even "improvements" to input signals may have arbitrary detrimental effects in the consuming system that are costly to diagnose and address. For example, consider the case in which an input signal was previously mis-calibrated. The model consuming it likely fit to these mis-calibrations, and a silent update that corrects the signal will have sudden ramifications for the model.

One common mitigation strategy for unstable data dependencies is to create a *versioned copy* of a given signal. For example, rather than allowing a semantic mapping of words to topic clusters to change over time, it might be reasonable to create a frozen version of this mapping and use it until such a time as an updated version has been fully vetted. Versioning carries its own costs, however, such as potential staleness and the cost to maintain multiple versions of the same signal over time.

**Underutilized Data Dependencies.** In code, underutilized dependencies are packages that are mostly unneeded [13]. Similarly, underutilized data dependencies are input signals that provide little incremental modeling benefit. These can make an ML system unnecessarily vulnerable to change, sometimes catastrophically so, even though they could be removed with no detriment.

As an example, suppose that to ease the transition from an old product numbering scheme to new product numbers, both schemes are left in the system as features. New products get only a new number, but old products may have both and the model continues to rely on the old numbers for some products. A year later, the code that stops populating the database with the old numbers is deleted. This will not be a good day for the maintainers of the ML system.

Underutilized data dependencies can creep into a model in several ways.

- **Legacy Features.** The most common case is that a feature $F$ is included in a model early in its development. Over time, $F$ is made redundant by new features but this goes undetected.
- **Bundled Features.** Sometimes, a group of features is evaluated and found to be beneficial. Because of deadline pressures or similar effects, all the features in the bundle are added to the model together, possibly including features that add little or no value.
- **$\epsilon$-Features.** As machine learning researchers, it is tempting to improve model accuracy even when the accuracy gain is very small or when the complexity overhead might be high.
- **Correlated Features.** Often two features are strongly correlated, but one is more directly causal. Many ML methods have difficulty detecting this and credit the two features equally, or may even pick the non-causal one. This results in brittleness if world behavior later changes the correlations.

Underutilized dependencies can be detected via exhaustive leave-one-feature-out evaluations. These should be run regularly to identify and remove unnecessary features.

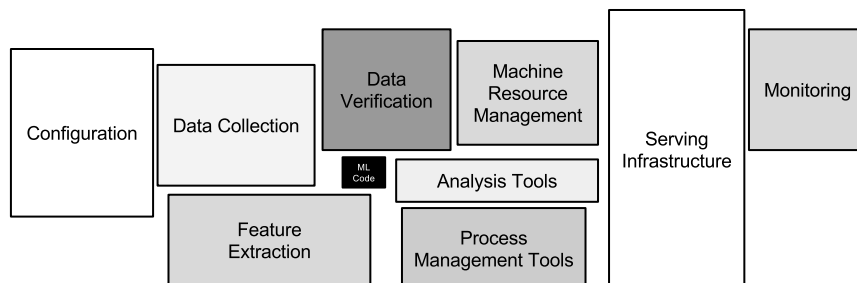

Figure 1: Only a small fraction of real-world ML systems is composed of the ML code, as shown by the small black box in the middle. The required surrounding infrastructure is vast and complex.

**Static Analysis of Data Dependencies.** In traditional code, compilers and build systems perform static analysis of dependency graphs. Tools for static analysis of data dependencies are far less common, but are essential for error checking, tracking down consumers, and enforcing migration and updates. One such tool is the automated feature management system described in [12], which enables data sources and features to be annotated. Automated checks can then be run to ensure that all dependencies have the appropriate annotations, and dependency trees can be fully resolved. This kind of tooling can make migration and deletion much safer in practice.

## 4   Feedback Loops

One of the key features of live ML systems is that they often end up influencing their own behavior if they update over time. This leads to a form of *analysis debt*, in which it is difficult to predict the behavior of a given model before it is released. These feedback loops can take different forms, but they are all more difficult to detect and address if they occur gradually over time, as may be the case when models are updated infrequently.

**Direct Feedback Loops.** A model may directly influence the selection of its own future training data. It is common practice to use standard supervised algorithms, although the theoretically correct solution would be to use bandit algorithms. The problem here is that bandit algorithms (such as contextual bandits [9]) do not necessarily scale well to the size of action spaces typically required for real-world problems. It is possible to mitigate these effects by using some amount of randomization [3], or by isolating certain parts of data from being influenced by a given model.

**Hidden Feedback Loops.** Direct feedback loops are costly to analyze, but at least they pose a statistical challenge that ML researchers may find natural to investigate [3]. A more difficult case is *hidden* feedback loops, in which two systems influence each other indirectly through the world.

One example of this may be if two systems independently determine facets of a web page, such as one selecting products to show and another selecting related reviews. Improving one system may lead to changes in behavior in the other, as users begin clicking more or less on the other components in reaction to the changes. Note that these hidden loops may exist between completely disjoint systems. Consider the case of two stock-market prediction models from two different investment companies. Improvements (or, more scarily, bugs) in one may influence the bidding and buying behavior of the other.

## 5   ML-System Anti-Patterns

It may be surprising to the academic community to know that only a tiny fraction of the code in many ML systems is actually devoted to learning or prediction – see Figure 1. In the language of Lin and Ryaboy, much of the remainder may be described as "plumbing" [11].

It is unfortunately common for systems that incorporate machine learning methods to end up with high-debt design patterns. In this section, we examine several system-design *anti-patterns* [4] that can surface in machine learning systems and which should be avoided or refactored where possible.

**Glue Code.**  ML researchers tend to develop general purpose solutions as self-contained packages. A wide variety of these are available as open-source packages at places like `mloss.org`, or from in-house code, proprietary packages, and cloud-based platforms.

Using generic packages often results in a *glue code* system design pattern, in which a massive amount of supporting code is written to get data into and out of general-purpose packages. Glue code is costly in the long term because it tends to freeze a system to the peculiarities of a specific package; testing alternatives may become prohibitively expensive. In this way, using a generic package can *inhibit* improvements, because it makes it harder to take advantage of domain-specific properties or to tweak the objective function to achieve a domain-specific goal. Because a mature system might end up being (at most) 5% machine learning code and (at least) 95% glue code, it may be less costly to create a clean native solution rather than re-use a generic package.

An important strategy for combating glue-code is to wrap black-box packages into common API's. This allows supporting infrastructure to be more reusable and reduces the cost of changing packages.

**Pipeline Jungles.**  As a special case of glue code, *pipeline jungles* often appear in data preparation. These can evolve organically, as new signals are identified and new information sources added incrementally. Without care, the resulting system for preparing data in an ML-friendly format may become a jungle of scrapes, joins, and sampling steps, often with intermediate files output. Managing these pipelines, detecting errors and recovering from failures are all difficult and costly [1]. Testing such pipelines often requires expensive end-to-end integration tests. All of this adds to technical debt of a system and makes further innovation more costly.

Pipeline jungles can only be avoided by thinking holistically about data collection and feature extraction. The clean-slate approach of scrapping a pipeline jungle and redesigning from the ground up is indeed a major investment of engineering effort, but one that can dramatically reduce ongoing costs and speed further innovation.

Glue code and pipeline jungles are symptomatic of integration issues that may have a root cause in overly separated "research" and "engineering" roles. When ML packages are developed in an ivory-tower setting, the result may appear like black boxes to the teams that employ them in practice. A hybrid research approach where engineers and researchers are embedded together on the same teams (and indeed, are often the same people) can help reduce this source of friction significantly [16].

**Dead Experimental Codepaths.**  A common consequence of glue code or pipeline jungles is that it becomes increasingly attractive in the short term to perform experiments with alternative methods by implementing experimental codepaths as conditional branches within the main production code. For any individual change, the cost of experimenting in this manner is relatively low—none of the surrounding infrastructure needs to be reworked. However, over time, these accumulated codepaths can create a growing debt due to the increasing difficulties of maintaining backward compatibility and an exponential increase in cyclomatic complexity. Testing all possible interactions between codepaths becomes difficult or impossible. A famous example of the dangers here was Knight Capital's system losing $465 million in 45 minutes, apparently because of unexpected behavior from obsolete experimental codepaths [15].

As with the case of *dead flags* in traditional software [13], it is often beneficial to periodically re-examine each experimental branch to see what can be ripped out. Often only a small subset of the possible branches is actually used; many others may have been tested once and abandoned.

**Abstraction Debt.**  The above issues highlight the fact that there is a distinct lack of strong abstractions to support ML systems. Zheng recently made a compelling comparison of the state ML abstractions to the state of database technology [17], making the point that nothing in the machine learning literature comes close to the success of the relational database as a basic abstraction. What is the right interface to describe a stream of data, or a model, or a prediction?

For distributed learning in particular, there remains a lack of widely accepted abstractions. It could be argued that the widespread use of Map-Reduce in machine learning was driven by the void of strong distributed learning abstractions. Indeed, one of the few areas of broad agreement in recent years appears to be that Map-Reduce is a poor abstraction for iterative ML algorithms.

The parameter-server abstraction seems much more robust, but there are multiple competing specifications of this basic idea [5, 10]. The lack of standard abstractions makes it all too easy to blur the lines between components.

**Common Smells.** In software engineering, a design *smell* may indicate an underlying problem in a component or system [7]. We identify a few ML system smells, not hard-and-fast rules, but as subjective indicators.

- **Plain-Old-Data Type Smell.** The rich information used and produced by ML systems is all to often encoded with plain data types like raw floats and integers. In a robust system, a model parameter should know if it is a log-odds multiplier or a decision threshold, and a prediction should know various pieces of information about the model that produced it and how it should be consumed.

- **Multiple-Language Smell.** It is often tempting to write a particular piece of a system in a given language, especially when that language has a convenient library or syntax for the task at hand. However, using multiple languages often increases the cost of effective testing and can increase the difficulty of transferring ownership to other individuals.

- **Prototype Smell.** It is convenient to test new ideas in small scale via prototypes. However, regularly relying on a prototyping environment may be an indicator that the full-scale system is brittle, difficult to change, or could benefit from improved abstractions and interfaces. Maintaining a prototyping environment carries its own cost, and there is a significant danger that time pressures may encourage a prototyping system to be used as a production solution. Additionally, results found at small scale rarely reflect the reality at full scale.

## 6  Configuration Debt

Another potentially surprising area where debt can accumulate is in the configuration of machine learning systems. Any large system has a wide range of configurable options, including which features are used, how data is selected, a wide variety of algorithm-specific learning settings, potential pre- or post-processing, verification methods, etc. We have observed that both researchers and engineers may treat configuration (and extension of configuration) as an afterthought. Indeed, verification or testing of configurations may not even be seen as important. In a mature system which is being actively developed, the number of lines of configuration can far exceed the number of lines of the traditional code. Each configuration line has a potential for mistakes.

Consider the following examples. Feature $A$ was incorrectly logged from 9/14 to 9/17. Feature $B$ is not available on data before 10/7. The code used to compute feature $C$ has to change for data before and after 11/1 because of changes to the logging format. Feature $D$ is not available in production, so a substitute features $D'$ and $D''$ must be used when querying the model in a live setting. If feature $Z$ is used, then jobs for training must be given extra memory due to lookup tables or they will train inefficiently. Feature $Q$ precludes the use of feature $R$ because of latency constraints.

All this messiness makes configuration hard to modify correctly, and hard to reason about. However, mistakes in configuration can be costly, leading to serious loss of time, waste of computing resources, or production issues. This leads us to articulate the following principles of good configuration systems:

- It should be easy to specify a configuration as a small change from a previous configuration.

- It should be hard to make manual errors, omissions, or oversights.

- It should be easy to see, visually, the difference in configuration between two models.

- It should be easy to automatically assert and verify basic facts about the configuration: number of features used, transitive closure of data dependencies, etc.

- It should be possible to detect unused or redundant settings.

- Configurations should undergo a full code review and be checked into a repository.

# 7 Dealing with Changes in the External World

One of the things that makes ML systems so fascinating is that they often interact directly with the external world. Experience has shown that the external world is rarely stable. This background rate of change creates ongoing maintenance cost.

**Fixed Thresholds in Dynamic Systems.** It is often necessary to pick a *decision threshold* for a given model to perform some action: to predict true or false, to mark an email as spam or not spam, to show or not show a given ad. One classic approach in machine learning is to choose a threshold from a set of possible thresholds, in order to get good tradeoffs on certain metrics, such as precision and recall. However, such thresholds are often manually set. Thus if a model updates on new data, the old manually set threshold may be invalid. Manually updating many thresholds across many models is time-consuming and brittle. One mitigation strategy for this kind of problem appears in [14], in which thresholds are learned via simple evaluation on heldout validation data.

**Monitoring and Testing.** Unit testing of individual components and end-to-end tests of running systems are valuable, but in the face of a changing world such tests are not sufficient to provide evidence that a system is working as intended. Comprehensive live monitoring of system behavior in real time combined with automated response is critical for long-term system reliability.

The key question is: what to monitor? Testable invariants are not always obvious given that many ML systems are intended to adapt over time. We offer the following starting points.

- **Prediction Bias.** In a system that is working as intended, it should usually be the case that the distribution of predicted labels is equal to the distribution of observed labels. This is by no means a comprehensive test, as it can be met by a null model that simply predicts average values of label occurrences without regard to the input features. However, it is a surprisingly useful diagnostic, and changes in metrics such as this are often indicative of an issue that requires attention. For example, this method can help to detect cases in which the world behavior suddenly changes, making training distributions drawn from historical data no longer reflective of current reality. Slicing prediction bias by various dimensions isolate issues quickly, and can also be used for automated alerting.

- **Action Limits.** In systems that are used to take actions in the real world, such as bidding on items or marking messages as spam, it can be useful to set and enforce action limits as a sanity check. These limits should be broad enough not to trigger spuriously. If the system hits a limit for a given action, automated alerts should fire and trigger manual intervention or investigation.

- **Up-Stream Producers.** Data is often fed through to a learning system from various up-stream producers. These up-stream processes should be thoroughly monitored, tested, and routinely meet a service level objective that takes the downstream ML system needs into account. Further any up-stream alerts must be propagated to the control plane of an ML system to ensure its accuracy. Similarly, any failure of the ML system to meet established service level objectives be also propagated down-stream to all consumers, and directly to their control planes if at all possible.

Because external changes occur in real-time, response must also occur in real-time as well. Relying on human intervention in response to alert pages is one strategy, but can be brittle for time-sensitive issues. Creating systems to that allow automated response without direct human intervention is often well worth the investment.

# 8 Other Areas of ML-related Debt

We now briefly highlight some additional areas where ML-related technical debt may accrue.

**Data Testing Debt.** If data replaces code in ML systems, and code should be tested, then it seems clear that some amount of testing of input data is critical to a well-functioning system. Basic sanity checks are useful, as more sophisticated tests that monitor changes in input distributions.

**Reproducibility Debt.**   As scientists, it is important that we can re-run experiments and get similar results, but designing real-world systems to allow for strict reproducibility is a task made difficult by randomized algorithms, non-determinism inherent in parallel learning, reliance on initial conditions, and interactions with the external world.

**Process Management Debt.**   Most of the use cases described in this paper have talked about the cost of maintaining a single model, but mature systems may have dozens or hundreds of models running simultaneously [14, 6]. This raises a wide range of important problems, including the problem of updating many configurations for many similar models safely and automatically, how to manage and assign resources among models with different business priorities, and how to visualize and detect blockages in the flow of data in a production pipeline. Developing tooling to aid recovery from production incidents is also critical. An important system-level smell to avoid are common processes with many manual steps.

**Cultural Debt.**   There is sometimes a hard line between ML research and engineering, but this can be counter-productive for long-term system health. It is important to create team cultures that reward deletion of features, reduction of complexity, improvements in reproducibility, stability, and monitoring to the same degree that improvements in accuracy are valued. In our experience, this is most likely to occur within heterogeneous teams with strengths in both ML research and engineering.

## 9   Conclusions: Measuring Debt and Paying it Off

Technical debt is a useful metaphor, but it unfortunately does not provide a strict metric that can be tracked over time. How are we to measure technical debt in a system, or to assess the full cost of this debt? Simply noting that a team is still able to move quickly is not in itself evidence of low debt or good practices, since the full cost of debt becomes apparent only over time. Indeed, moving quickly often *introduces* technical debt. A few useful questions to consider are:

- How easily can an entirely new algorithmic approach be tested at full scale?
- What is the transitive closure of all data dependencies?
- How precisely can the impact of a new change to the system be measured?
- Does improving one model or signal degrade others?
- How quickly can new members of the team be brought up to speed?

We hope that this paper may serve to encourage additional development in the areas of maintainable ML, including better abstractions, testing methodologies, and design patterns. Perhaps the most important insight to be gained is that technical debt is an issue that engineers and researchers both need to be aware of. Research solutions that provide a tiny accuracy benefit at the cost of massive increases in system complexity are rarely wise practice. Even the addition of one or two seemingly innocuous data dependencies can slow further progress.

Paying down ML-related technical debt requires a specific commitment, which can often only be achieved by a shift in team culture. Recognizing, prioritizing, and rewarding this effort is important for the long term health of successful ML teams.

**Acknowledgments**

This paper owes much to the important lessons learned day to day in a culture that values both innovative ML research and strong engineering practice. Many colleagues have helped shape our thoughts here, and the benefit of accumulated folk wisdom cannot be overstated. We would like to specifically recognize the following: Roberto Bayardo, Luis Cobo, Sharat Chikkerur, Jeff Dean, Philip Henderson, Arnar Mar Hrafnkelsson, Ankur Jain, Joe Kovac, Jeremy Kubica, H. Brendan McMahan, Satyaki Mahalanabis, Lan Nie, Michael Pohl, Abdul Salem, Sajid Siddiqi, Ricky Shan, Alan Skelly, Cory Williams, and Andrew Young.

A short version of this paper was presented at the SE4ML workshop in 2014 in Montreal, Canada.

# References

[1] R. Ananthanarayanan, V. Basker, S. Das, A. Gupta, H. Jiang, T. Qiu, A. Reznichenko, D. Ryabkov, M. Singh, and S. Venkataraman. Photon: Fault-tolerant and scalable joining of continuous data streams. In *SIGMOD '13: Proceedings of the 2013 international conference on Management of data*, pages 577–588, New York, NY, USA, 2013.

[2] A. Anonymous. Machine learning: The high-interest credit card of technical debt. *SE4ML: Software Engineering for Machine Learning (NIPS 2014 Workshop)*.

[3] L. Bottou, J. Peters, J. Quiñonero Candela, D. X. Charles, D. M. Chickering, E. Portugaly, D. Ray, P. Simard, and E. Snelson. Counterfactual reasoning and learning systems: The example of computational advertising. *Journal of Machine Learning Research*, 14(Nov), 2013.

[4] W. J. Brown, H. W. McCormick, T. J. Mowbray, and R. C. Malveau. Antipatterns: refactoring software, architectures, and projects in crisis. 1998.

[5] T. M. Chilimbi, Y. Suzue, J. Apacible, and K. Kalyanaraman. Project adam: Building an efficient and scalable deep learning training system. In *11th USENIX Symposium on Operating Systems Design and Implementation, OSDI '14, Broomfield, CO, USA, October 6-8, 2014.*, pages 571–582, 2014.

[6] B. Dalessandro, D. Chen, T. Raeder, C. Perlich, M. Han Williams, and F. Provost. Scalable hands-free transfer learning for online advertising. In *Proceedings of the 20th ACM SIGKDD international conference on Knowledge discovery and data mining*, pages 1573–1582. ACM, 2014.

[7] M. Fowler. Code smells. `http://http://martinfowler.com/bliki/CodeSmell.html`.

[8] M. Fowler. *Refactoring: improving the design of existing code*. Pearson Education India, 1999.

[9] J. Langford and T. Zhang. The epoch-greedy algorithm for multi-armed bandits with side information. In *Advances in neural information processing systems*, pages 817–824, 2008.

[10] M. Li, D. G. Andersen, J. W. Park, A. J. Smola, A. Ahmed, V. Josifovski, J. Long, E. J. Shekita, and B. Su. Scaling distributed machine learning with the parameter server. In *11th USENIX Symposium on Operating Systems Design and Implementation, OSDI '14, Broomfield, CO, USA, October 6-8, 2014.*, pages 583–598, 2014.

[11] J. Lin and D. Ryaboy. Scaling big data mining infrastructure: the twitter experience. *ACM SIGKDD Explorations Newsletter*, 14(2):6–19, 2013.

[12] H. B. McMahan, G. Holt, D. Sculley, M. Young, D. Ebner, J. Grady, L. Nie, T. Phillips, E. Davydov, D. Golovin, S. Chikkerur, D. Liu, M. Wattenberg, A. M. Hrafnkelsson, T. Boulos, and J. Kubica. Ad click prediction: a view from the trenches. In *The 19th ACM SIGKDD International Conference on Knowledge Discovery and Data Mining, KDD 2013, Chicago, IL, USA, August 11-14, 2013*, 2013.

[13] J. D. Morgenthaler, M. Gridnev, R. Sauciuc, and S. Bhansali. Searching for build debt: Experiences managing technical debt at google. In *Proceedings of the Third International Workshop on Managing Technical Debt*, 2012.

[14] D. Sculley, M. E. Otey, M. Pohl, B. Spitznagel, J. Hainsworth, and Y. Zhou. Detecting adversarial advertisements in the wild. In *Proceedings of the 17th ACM SIGKDD International Conference on Knowledge Discovery and Data Mining, San Diego, CA, USA, August 21-24, 2011*, 2011.

[15] Securities and E. Commission. *SEC Charges Knight Capital With Violations of Market Access Rule*, 2013.

[16] A. Spector, P. Norvig, and S. Petrov. Google's hybrid approach to research. *Communications of the ACM*, 55 Issue 7, 2012.

[17] A. Zheng. The challenges of building machine learning tools for the masses. *SE4ML: Software Engineering for Machine Learning (NIPS 2014 Workshop)*.

